# Markov Models for Automated ECG Interval Analysis

**Nicholas P. Hughes, Lionel Tarassenko and Stephen J. Roberts**
Department of Engineering Science
University of Oxford
Oxford, 0X1 3PJ, UK
{*nph,lionel,sjrob*}*@robots.ox.ac.uk*

## Abstract

We examine the use of hidden Markov and hidden semi-Markov models for automatically segmenting an electrocardiogram waveform into its constituent waveform features. An undecimated wavelet transform is used to generate an overcomplete representation of the signal that is more appropriate for subsequent modelling. We show that the state durations implicit in a standard hidden Markov model are ill-suited to those of real ECG features, and we investigate the use of hidden semi-Markov models for improved state duration modelling.

## 1 Introduction

The development of new drugs by the pharmaceutical industry is a costly and lengthy process, with the time from concept to final product typically lasting ten years. Perhaps the most critical stage of this process is the *phase one* study, where the drug is administered to humans for the first time. During this stage each subject is carefully monitored for any unexpected adverse effects which may be brought about by the drug. Of particular interest is the electrocardiogram (ECG[1]) of the patient, which provides detailed information about the state of the patient's heart.

By examining the ECG signal in detail it is possible to derive a number of informative measurements from the characteristic ECG waveform. These can then be used to assess the medical well-being of the patient, and more importantly, detect any potential side effects of the drug on the cardiac rhythm. The most important of these measurements is the "QT interval". In particular, drug-induced prolongation of the QT interval (so called Long QT Syndrome) can result in a very fast, abnormal heart rhythm known as *torsade de pointes*, which is often followed by sudden cardiac death [2].

In practice, QT interval measurements are carried out manually by specially trained ECG analysts. This is an expensive and time consuming process, which is susceptible to mistakes by the analysts and provides no associated degree of confidence (or accuracy) in the measurements. This problem was recently highlighted in the case of the antihistamine

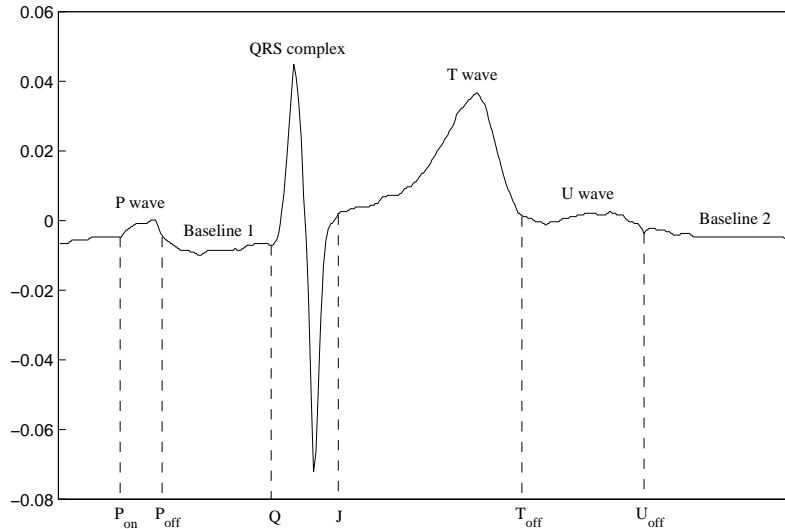

Figure 1: A human ECG waveform.

*terfenadine*, which had the side-effect of significantly prolonging the QT interval in a number of patients. Unfortunately this side-effect was not detected in the clinical trials and only came to light after a large number of people had unexpectedly died whilst taking the drug [8].

In this paper we consider the problem of automated ECG interval analysis from a machine learning perspective. In particular, we examine the use of hidden Markov models for automatically segmenting an ECG signal into its constituent waveform features. A redundant wavelet transform is used to provide an informative representation which is both robust to noise and tuned to the morphological characteristics of the waveform features. Finally we investigate the use of hidden semi-Markov models for explicit state duration modelling.

## 2 The Electrocardiogram

### 2.1 The ECG Waveform

Each individual heartbeat is comprised of a number of distinct cardiological stages, which in turn give rise to a set of distinct features in the ECG waveform. These features represent either *depolarization* (electrical discharging) or *repolarization* (electrical recharging) of the muscle cells in particular regions of the heart. Figure 1 shows a human ECG waveform and the associated features. The standard features of the ECG waveform are the P wave, the QRS complex and the T wave. Additionally a small U wave (following the T wave) is occasionally present.

The cardiac cycle begins with the P wave (the start and end points of which are referred to as $P_{on}$ and $P_{off}$), which corresponds to the period of *atrial depolarization* in the heart. This is followed by the QRS complex, which is generally the most recognisable feature of an ECG waveform, and corresponds to the period of *ventricular depolarization*. The start and end points of the QRS complex are referred to as the Q and J points. The T wave follows the QRS complex and corresponds to the period of *ventricular repolarization*. The end point of the T wave is referred to as $T_{off}$ and represents the end of the cardiac cycle (presuming the absence of a U wave).

## 2.2 ECG Interval Analysis

The timing between the onset and offset of particular features of the ECG (referred to as an *interval*) is of great importance since it provides a measure of the state of the heart and can indicate the presence of certain cardiological conditions. The two most important intervals in the ECG waveform are the QT interval and the PR interval. The QT interval is defined as the time from the *start* of the QRS complex to the *end* of the T wave, i.e. $T_{off} - Q$, and corresponds to the total duration of electrical activity (both depolarization and repolarization) in the ventricles. Similarly, the PR interval is defined as the time from the *start* of the P wave to the *start* of the QRS complex, i.e. $Q - P_{on}$, and corresponds to the time from the onset of atrial depolarization to the onset of ventricular depolarization.

The measurement of the QT interval is complicated by the fact that a precise mathematical definition of the end of the T wave does not exist. Thus T wave end measurements are inherently subjective and the resulting QT interval measurements often suffer from a high degree of inter- and intra-analyst variability. An automated ECG interval analysis system, which could provide robust and consistent measurements (together with an associated degree of confidence in each measurement), would therefore be of great benefit to the medical community.

## 2.3 Previous Work on Automated ECG Interval Analysis

The vast majority of algorithms for automated QT analysis are based on threshold methods which attempt to predict the end of the T wave as the point where the T wave crosses a predetermined threshold [3]. An exception to this is the work of Koski [4] who trained a hidden Markov model on raw ECG data using the Baum-Welch algorithm. However the performance of this model was not assessed against a labelled data set of ECG waveforms. More recently, Graja and Boucher have investigated the use of hidden Markov tree models for segmenting ECG signals encoded with the discrete wavelet transform [2].

# 3  Data Collection

In order to develop an automated system for ECG interval analysis, we collected a data set of over 100 ECG waveforms (sampled at 500 Hz), together with the corresponding waveform feature boundaries[3] as determined by a group of expert ECG analysts. Due to time constraints it was not possible for each expert analyst to label every ECG waveform in the data set. Therefore we chose to distribute the waveforms at random amongst the different experts (such that each waveform was measured by one expert only).

For each ECG waveform, the following points were labelled: $P_{on}$, $P_{off}$, Q, J and $T_{off}$ (if a U wave was present the $U_{off}$ point was also labelled). In addition, the point corresponding to the start of the next P wave (i.e. the P wave of the following heart beat), $NP_{on}$, was also labelled. During the data collection exercise, we found that it was not possible to obtain reliable estimates for the $T_{on}$ and $U_{on}$ points, and therefore these were taken to be the J and $T_{off}$ points respectively.

# 4  A Hidden Markov Model for ECG Interval Analysis

It is natural to view the ECG signal as the result of a generative process, in which each waveform feature is *generated* by the corresponding cardiological state of the heart. In addition, the ECG state sequence obeys the Markov property, since each state is solely

| | | | | | |
|---|---|---|---|---|---|
| P wave | **5.5** | 47.2 | 0.5 | 4.4 | 26.5 | 15.9 |
| Baseline 1 | 1.7 | **80.0** | 1.6 | 1.3 | 9.5 | 5.9 |
| QRS complex | 1.0 | 11.3 | **79.0** | 4.6 | 2.7 | 1.4 |
| T wave | 0.9 | 1.8 | 1.2 | **83.6** | 7.3 | 5.2 |
| Baseline 2 | 2.3 | 32.2 | 1.3 | 3.5 | **31.8** | 28.9 |
| U wave | 0.6 | 25.3 | 0.6 | 3.9 | 26.8 | **42.8** |

Table 1: Percentage confusion matrix for an HMM trained on the raw ECG data.

dependent on the previous state. Thus, hidden Markov models (HMMs) would seem ideally suited to the task of segmenting an ECG signal into its constituent waveform features.

Using the labelled data set of ECG waveforms we trained a hidden Markov model in a *supervised* manner. The model was comprised of the following states: P wave, QRS complex, T wave, U wave, and Baseline. The parameters of the transition matrix $a_{ij}$ were computed using the maximum likelihood estimates, given by:

$$\hat{a}_{ij} = n_{ij}/\sum_k n_{ik} \qquad (1)$$

where $n_{ij}$ is the total number of transitions from state $i$ to state $j$ over all of the label sequences. We estimated the observation (or emission) probability densities $b_i$ for each state $i$ by fitting a Gaussian mixture model (GMM) to the set of signal samples corresponding to that particular state[4]. Model selection for the GMM was performed using the minimum description length framework [1].

In our initial experiments, we found that the use of a single state to represent all the regions of baseline in the ECG waveform resulted in poor performance when the model was used to infer the underlying state sequence of new unseen waveforms. In particular, a single baseline state allowed for the possibility of the model returning to the P wave state, following a P wave - Baseline sequence. Therefore we decided to partition the Baseline state into two separate states; one corresponding to the region of baseline between the $P_{off}$ and Q points (which we termed "Baseline 1"), and a second corresponding to the region between the $T_{off}$ and $NP_{on}$ points[5] (termed "Baseline 2").

In order to fully evaluate the performance of our model, we performed 5-fold cross-validation on the data set of 100 labelled ECGs. Prior to training and testing, the raw ECG data was pre-processed to have zero mean and unit energy. This was done in order to normalise the dynamic range of the signals and stabilise the baseline sections. Once the model had been trained, the Viterbi algorithm [9] was used to infer the optimal state sequence for each of the signals in the test set.

Table 1 shows the resulting confusion matrix (computed from the state assignments on a *sample-point* basis). Although reasonable classification accuracies are obtained for the QRS complex and T wave states, the P wave state is almost entirely misclassified as Baseline 1, Baseline 2 or U wave. In order to improve the performance of the model, we require an encoding of the ECG that captures the key temporal and spectral characteristics of the waveform features in a more informative representation than that of the raw time series data alone. Thus we now examine the use of wavelet methods for this purpose.

| P wave | **74.2** | 14.4 | 0.1 | 0.3 | 11.0 | 0 |
|---|---|---|---|---|---|---|
| Baseline 1 | 15.8 | **81.5** | 1.7 | 0.1 | 0.9 | 0 |
| QRS complex | 0 | 2.1 | **94.4** | 3.5 | 0 | 0 |
| T wave | 0 | 0 | 1.0 | **96.1** | 2.2 | 0.7 |
| Baseline 2 | 1.4 | 0 | 0 | 1.6 | **95.6** | 1.4 |
| U wave | 0.1 | 0.1 | 0.1 | 1.7 | 85.6 | **12.4** |

Table 2: Percentage confusion matrix for an HMM trained on the wavelet encoded ECG.

### 4.1 Wavelet Encoding of ECG

Wavelets are a class of functions that possess compact support and form a basis for all finite energy signals. They are able to capture the non-stationary spectral characteristics of a signal by decomposing it over a set of *atoms* which are localised in both time and frequency. These atoms are generated by scaling and translating a single mother wavelet.

The most popular wavelet transform algorithm is the discrete wavelet transform (DWT), which uses the set of *dyadic* scales (i.e. those based on powers of two) and translates of the mother wavelet to form an orthonormal basis for signal analysis. The DWT is therefore most suited to applications such as data compression where a compact description of a signal is required. An alternative transform is derived by allowing the translation parameter to vary continuously, whilst restricting the scale parameter to a dyadic scale (thus, the set of time-frequency atoms now forms a *frame*). This leads to the undecimated wavelet transform[6] (UWT), which for a signal $s \in \mathbf{L}^2(\mathbb{R})$, is given by:

$$w_v(\tau) = \frac{1}{\sqrt{v}} \int_{-\infty}^{+\infty} s(t)\, \psi^* \left( \frac{t - \tau}{v} \right) dt \qquad v = 2^k, k \in \mathbb{Z}, \tau \in \mathbb{R} \qquad (2)$$

where $w_v(\tau)$ are the UWT coefficients at scale $v$ and shift $\tau$, and $\psi^*$ is the complex conjugate of the mother wavelet. In practice the UWT can be computed in $O(N \log N)$ using fast filter bank algorithms [6].

The UWT is particularly well-suited to ECG interval analysis as it provides a time-frequency description of the ECG signal on a sample-by-sample basis. In addition, the UWT coefficients are translation-invariant (unlike the DWT coefficients), which is important for pattern recognition applications.

In order to find the most effective wavelet basis for our application, we examined the performance of HMMs trained on ECG data encoded with wavelets from the Daubechies, Symlet, Coiflet and Biorthogonal wavelet families. In the frequency domain, a wavelet at a given scale is associated with a bandpass filter[7] of a particular centre frequency. Thus the optimal wavelet basis will correspond to the set of bandpass filters that are tuned to the unique spectral characteristics of the ECG.

In our experiments we found that the Coiflet wavelet with two vanishing moments resulted in the highest overall classification accuracy. Table 2 shows the results for this wavelet. It is evident that the UWT encoding results in a significant improvement in classification accuracy (for all but the U wave state), when compared with the results obtained on the raw ECG data.

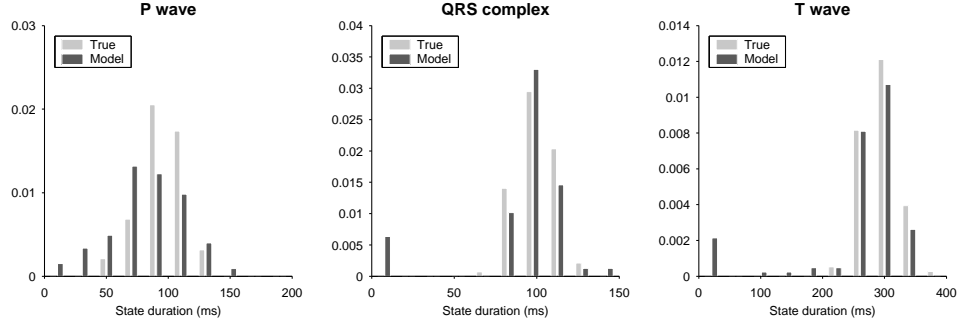

Figure 2: Histograms of the true state durations and those decoded by the HMM.

### 4.2 HMM State Durations

A significant limitation of the standard hidden Markov model is the manner in which it models state durations. For a given state $i$ with self-transition coefficient $a_{ii}$, the probability density of the state duration $d$ is a geometric distribution, given by:

$$p_i(d) = (a_{ii})^{d-1}(1 - a_{ii}) \qquad (3)$$

For the waveform features of the ECG signal, this geometric distribution is inappropriate. Figure 2 shows histograms of the true state durations and the durations of the states decoded by the HMM, for each of the P wave, QRS complex and T wave states. In each case it is clear that a significant number of decoded states have a duration that is much shorter than the minimum state duration observed with real ECG signals. Thus for a given ECG waveform the decoded state sequence may contain many more state transitions than are actually present in the signal. The resulting HMM state segmentation is then likely to be poor and the resulting QT and PR interval measurements unreliable.

One solution to this problem is to post-process the decoded state sequences using a median filter designed to smooth out sequences whose duration is known to be physiologically implausible. A more principled and more effective approach, however, is to model the probability density of the individual state durations *explicitly*, using a hidden semi-Markov model.

## 5 A Hidden Semi-Markov Model for ECG Interval Analysis

A hidden semi-Markov model (HSMM) differs from a standard HMM in that each of the self-transition coefficients $a_{ii}$ are set to zero, and an explicit probability density is specified for the duration of each state [5]. In this way, the individual state duration densities govern the amount of time the model spends in a given state, and the transition matrix governs the probability of the next state once this time has elapsed. Thus the underlying stochastic process is now a "semi-Markov" process.

To model the durations $p_i(d)$ of the various waveform features of the ECG, we used a Gamma density since this is a positive distribution which is able to capture the inherent skewness of the ECG state durations. For each state $i$, maximum likelihood estimates of the shape and scale parameters were computed directly from the set of labelled ECG signals (as part of the cross-validation procedure).

In order to infer the most probable state sequence $Q = \{q_1 q_2 \cdots q_T\}$ for a given observation sequence $O = \{O_1 O_2 \cdots O_T\}$, the standard Viterbi algorithm must be modified to

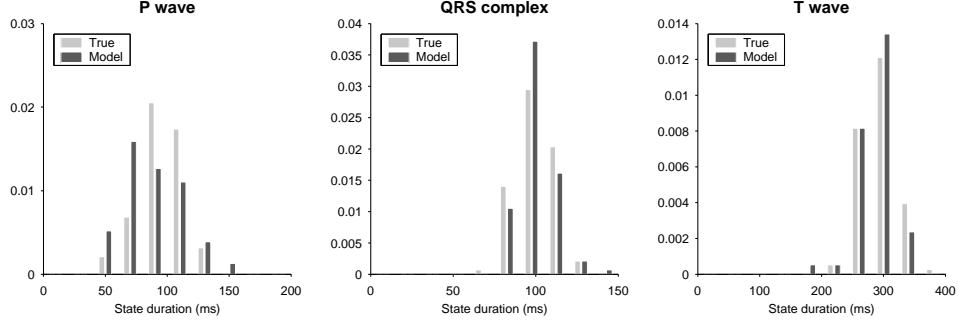

Figure 3: Histograms of the true state durations and those decoded by the HSMM.

handle the explicit state duration densities of the HSMM. We start by defining the likelihood of the most probable state sequence that accounts for the first $t$ observations and ends in state $i$:

$$\delta_t(i) = \max_{q_1 q_2 \cdots q_{t-1}} p(q_1 q_2 \cdots q_t = i, O_1 O_2 \cdots O_t | \lambda) \qquad (4)$$

where $\lambda$ is the set of parameters governing the HSMM. The recurrence relation for computing $\delta_t(i)$ is then given by:

$$\delta_t(i) = \max_{d_i} \Big\{ \max_j \big\{ \delta_{t-d_i}(j) a_{ji} \big\} p_i(d_i) \, \Pi_{t'=t-d_i+1}^{t} b_i(O_{t'}) \Big\} \qquad (5)$$

where the outer maximisation is performed over all possible values of the state duration $d_i$ for state $i$, and the inner maximisation is over all states $j$. At each time $t$ and for each state $i$, the two arguments that maximise equation (5) are recorded, and a simple backtracking procedure can then be used to find the most probable state sequence.

The time complexity of the Viterbi decoding procedure for an HSMM is given by $O(K^2 \, T \, D_{max})$, where $K$ is the total number of states, and $D_{max}$ is the maximum range of state durations over all $K$ states, i.e. $D_{max} = \max_i (\max(d_i) - \min(d_i))$. As noted in [7], scaling the computation of $\delta_t(i)$ to avoid underflow is non-trivial. However, by simply computing $\log \delta_t(i)$ it is possible to avoid any numerical problems.

Figure 3 shows histograms of the resulting state durations for an HSMM trained on a wavelet encoding of the ECG (using 5-fold cross-validation). Clearly, the durations of the decoded state sequences are very well matched to the true durations of each of the ECG features. This improvement in duration modelling is reflected in the accuracy and robustness of the segmentations produced by the HSMM.

| Model | $P_{on}$ | Q | J | $T_{off}$ |
|---|---|---|---|---|
| HMM on raw ECG | 157 | 31 | 27 | 139 |
| HMM on wavelet encoded ECG | 12 | 11 | 20 | 46 |
| HSMM on wavelet encoded ECG | 13 | 3 | 7 | 12 |

Table 3: Mean absolute segmentation errors (in milliseconds) for each of the models.

Table 3 shows the mean absolute errors[8] for the $P_{on}$, Q, J and $T_{off}$ points, for each of the models discussed. On the important task of accurately determining the Q and $T_{off}$ points for QT interval measurements, the HSMM significantly outperforms the HMM.

## 6   Discussion

In this work we have focused on the two core issues in developing an automated system for ECG interval analysis: the choice of representation for the ECG signal and the choice of model for the segmentation. We have demonstrated that wavelet methods, and in particular the *undecimated* wavelet transform, can be used to generate an encoding of the ECG which is tuned to the unique spectral characteristics of the ECG waveform features. With this representation the performance of the models on new unseen ECG waveforms is significantly better than similar models trained on the raw time series data. We have also shown that the robustness of the segmentation process can be improved through the use of explicit state duration modelling with hidden semi-Markov models. With these models the detection accuracy of the $Q$ and $T_{off}$ points compares favourably with current methods for automated QT analysis [3, 2].

A key advantage of probabilistic models over traditional threshold-based methods for ECG segmentation is that they can be used to generate a confidence measure for each segmented ECG signal. This is achieved by considering the log likelihood of the observed signal given the model, i.e. $\log p(O|\lambda)$, which can be computed efficiently for both HMMs and HSMMs. Given this confidence measure, it should be possible to determine a suitable threshold for rejecting ECG signals which are either too noisy or too corrupted to provide reliable estimates of the QT and PR intervals. The robustness with which we can detect such unreliable QT interval measurements based on this log likelihood score is one of the main focuses of our current research.

### Acknowledgements

We thank Cardio Analytics Ltd for help with data collection and labelling, and Oxford BioSignals Ltd for funding this research. NH thanks Iead Rezek for many useful discussions, and the anonymous reviewers for their helpful comments.

## Footnotes

[1]The ECG is also referred to as the EKG.

[2]This is known as Sudden Arrhythmia Death Syndrome, or SADS.

[3]We developed a novel software application which enabled an ECG analyst to label the boundaries of each of the features of an ECG waveform, using a pair of "onscreen calipers".

[4]We also investigated *autoregressive* observation densities, although these were found to perform poorly in comparison to GMMs.

[5]If a U wave was present the $U_{off}$ point was used instead of $T_{off}$.

[6] The undecimated wavelet transform is also known as the stationary wavelet transform and the translation-invariant wavelet transform.

[7] These filters satisfy a constant relative bandwidth property, known as "constant-$Q$".

[8]The error was taken to be the time difference from the first decoded segment boundary to the true segment boundary (of the same type).

## References

[1] M. A. T. Figueiredo and A. K. Jain. Unsupervised learning of finite mixture models. *IEEE Transactions on Pattern Analysis and Machine Intelligence*, 24(3):381–396, 2002.

[2] S. Graja and J. M. Boucher. Multiscale hidden Markov model applied to ECG segmentation. In *WISP 2003: IEEE International Symposium on Intelligent Signal Processing*, pages 105–109, Budapest, Hungary, 2003.

[3] R. Jané, A. Blasi, J. García, and P. Laguna. Evaluation of an automatic threshold based detector of waveform limits in Holter ECG with QT database. In *Computers in Cardiology*, pages 295–298. IEEE Press, 1997.

[4] A. Koski. Modelling ECG signals with hidden Markov models. *Artificial Intelligence in Medicine*, 8:453–471, 1996.

[5] S. E. Levinson. Continuously variable duration hidden Markov models for automatic speech recognition. *Computer Speech and Language*, 1(1):29–45, 1986.

[6] S. Mallat. *A Wavelet Tour of Signal Processing*. Academic Press, 2nd edition, 1999.

[7] K. P. Murphy. Hidden semi-Markov models. Technical report, MIT AI Lab, 2002.

[8] C. M. Pratt and S. Ruberg. The dose-response relationship between Terfenadine (Seldane) and the QTc interval on the scalar electrocardiogram in normals and patients with cardiovascular disease and the QTc interval variability. *American Heart Journal*, 131(3):472–480, 1996.

[9] L. R. Rabiner. A tutorial on hidden Markov models and selected applications in speech recognition. *Proceedings of the IEEE*, 77(2):257–286, 1989.
